# Optimal Asset Allocation
## using
# Adaptive Dynamic Programming

**Ralph Neuneier***

Siemens AG, Corporate Research and Development

Otto-Hahn-Ring 6, D-81730 München, Germany

## Abstract

In recent years, the interest of investors has shifted to computer-ized asset allocation (portfolio management) to exploit the growing dynamics of the capital markets. In this paper, asset allocation is formalized as a *Markovian Decision Problem* which can be opti-mized by applying *dynamic programming* or reinforcement learning based algorithms. Using an artificial exchange rate, the asset allo-cation strategy optimized with reinforcement learning (*Q-Learning*) is shown to be equivalent to a policy computed by dynamic pro-gramming. The approach is then tested on the task to invest liquid capital in the German stock market. Here, neural networks are used as value function approximators. The resulting asset alloca-tion strategy is superior to a heuristic benchmark policy. This is a further example which demonstrates the applicability of neural network based reinforcement learning to a problem setting with a high dimensional state space.

## 1  Introduction

Billions of dollars are daily pushed through the international capital markets while brokers shift their investments to more promising assets. Therefore, there is a great interest in achieving a deeper understanding of the capital markets and in developing efficient tools for exploiting the dynamics of the markets.

Asset allocation (portfolio management) is the investment of liquid capital to various trading opportunities like stocks, futures, foreign exchanges and others. A portfolio is constructed with the aim of achieving a maximal expected return for a given risk level and time horizon. To compose an optimal portfolio, the investor has to solve a difficult optimization problem consisting of two phases (Brealy, 1991). First, the expected yields are estimated simultaneously with a certainty measure. Second, based on these estimates, a portfolio is constructed obeying the risk level the investor is willing to accept (*mean-variance* techniques). The problem is further complicated if transaction costs must be considered and if the investor wants to revise the decision at every time step. In recent years, neural networks (NN) have been successfully used for the first task. Typically, a NN delivers the expected future values of a time series based on data of the past. Furthermore, a confidence measure which expresses the certainty of the prediction is provided.

In the following, the modeling phase and the search for an optimal portfolio are combined and embedded in the framework of *Markovian Decision Problems*, MDP. That theory formalizes control problems within stochastic environments (Bertsekas, 1987, Elton, 1971). If the discrete state space is small and if an accurate model of the system is available, MDP can be solved by conventional *Dynamic Programming*, DP. On the other extreme, reinforcement learning methods, e.g. *Q-Learning*, QL, can be applied to problems with large state spaces and with no appropriate model available (Singh, 1994).

## 2  Portfolio Management is a Markovian Decision Problem

The following simplifications do not restrict the generalization of the proposed methods with respect to real applications but will help to clarify the relationship between MDP and portfolio optimization.

- There is only one possible asset for a Deutsch-Mark based investor, say a foreign currency called Dollar, US-$.

- The investor is small and does not influence the market by her/his trading.

- The investor has no risk aversion and always invests the total amount.

- The investor may trade at each time step for an infinite time horizon.

MDP provide a model for multi-stage decision making problems in stochastic environments. MDP can be described by a finite state set $S = 1, \ldots, n$, a finite set $U(i)$ of admissible control actions for every state $i \in S$, a set of transition probabilities $p_{ij}^{\pi}$, which describe the dynamics of the system, and a return function[1] $r(i, j, u(i))$, with $i, j \in S, u(i) \in U(i)$. Furthermore, there is a stationary policy $\pi(i)$, which delivers for every state an admissible action $u(i)$. One can compute the value-function $V_i^{\pi}$ of a given state and policy,

$$V_{i_0}^{\pi} = E[\sum_{t=0}^{\infty} \gamma^t R(i_t, \pi(i_t))], \tag{1}$$

where $E$ indicates the expected value, $\gamma$ is the discount factor with $0 \leq \gamma < 1$, and where $R$ are the expected returns, $R = E_j(r(i, j, u(i))$. The aim is now to find a policy $\pi^*$ with the optimal value-function $V_i^* = \max_\pi V_i^\pi$ for all states.

In the context discussed here, a state vector consists of elements which describe the financial time series, and of elements which quantify the current value of the investment. For the simple example above, the state vector is the triple of the exchange rate, $x_t$, the wealth of the portfolio, $c_t$, expressed in the basis currency (here DM), and a binary variable $b$, representing the fact that currently the investment is in DM or US-\$.

Note, that out of the variables which form the state vector, the exchange rate is actually independent of the portfolio decisions, but the wealth and the returns are not. Therefore, asset allocation is a control problem and may not be reduced to pure prediction.[2] This problem has the attractive feature that, because the investments do not influence the exchange rate, we do not need to invest real money during the training phase of QL until we are convinced that our strategy works.

## 3 Dynamic Programming: Off-line and Adaptive

The optimal value function $V^*$ is the unique solution of the well-known Bellman equation (Bertsekas, 1987). According to that equation one has to maximize the expected return for the next step and follow an optimal policy thereafter in order to achieve global optimal behavior (Bertsekas, 1987). An optimal policy can be easily derived from $V^*$ by choosing a $\pi(i)$ which satisfies the Bellman equation. For nonlinear systems and non-quadric cost functions, $V^*$ is typically found by using an iterative algorithm, *value iteration*, which converges asymptotically to $V^*$. Value iteration applies repeatedly the operator $T$ for all states $i$,

$$T_i(V) = \max_{u(i) \in U(i)} \left( R(i, u(i)) + \gamma \sum_{j \in S} p_{ij}^\pi V_j \right). \tag{2}$$

Value iteration assumes that the expected return function $R(i, u(i))$ and the transition probabilities $p_{ij}^\pi$ (i. e. the model) are known. *Q-Learning*, (QL), is a *reinforcement-learning* method that does not require a model of the system but optimizes the policy by sampling state-action pairs and returns while interacting with the system (Barto, 1989). Let's assume that the investor executes action $u(i)$ at state $i$, and that the system moves to a new state $j$. Let $r(i, j, u(i))$ denote the actual return. QL then uses the update equation

$$\begin{aligned} Q(i, u(i)) &= (1 - \eta)Q(i, u(i)) + \eta(r(i, j, u(i)) + \gamma \max_{u(j)} Q(j, u(j))) \\ Q(k, v) &= Q(k, v), \text{ for all } k \neq i \text{ and } v \neq u(i) \end{aligned} \tag{3}$$

where $\eta$ is the learning rate and $Q(i, u(i))$ are the tabulated Q-values. One can prove, that this relaxation algorithm converges (under some conditions) to the optimal Q-values (Singh, 1994).

The selection of the action $u(i)$ should be guided by the trade-off between exploration and exploitation. In the beginning, the actions are typically chosen randomly (exploration) and in the course of training, actions with larger Q-values are chosen with increasingly higher probability (exploitation). The implementation in the following experiments is based on the Boltzmann-distribution using the actual Q-values and a slowly decreasing temperature parameter (see Barto, 1989).

## 4  Experiment I: Artificial Exchange Rate

In this section we use an exchange-rate model to demonstrate how DP and Q-Learning can be used to optimize asset allocation.

The artificial exchange rate $x_t$ is in the range between 1 and 2 representing the value of 1 US-\$ in DM. The transition probabilities $p_{ij}$ of the exchange rate are chosen to simulate a situation where the $x_t$ follows an increasing trend, but with higher values of $x_t$, a drop to very low values becomes more and more probable. A realization of the time series is plotted in the upper part of fig. 2. The random state variable $c_t$ depends on the investor's decisions $u_t$, and is further influenced by $x_t$, $x_{t+1}$, and $c_{t-1}$. A complete state vector consists of the current exchange rate $x_t$ and the capital $c_t$, which is always calculated in the basis currency (DM). Its sign represents the actual currency, i. e., $c_t = -1.2$ stands for an investment in US-\$ worth of 1.2 DM, and $c_t = 1.2$ for a capital of 1.2 DM. $c_t$ and $x_t$ are discretized in 10 bins each. The transaction costs $\xi = 0.1 + |c/100|$ are a combination of fixed $(0.1)$ and variable costs $(|c/100|)$. Transactions only apply, if the currency is changed from DM to US-\$. The immediate return $r_t(x_t, c_t, x_{t+1}, u_t)$ is computed as in table 1. If the decision has been made to change the portfolio into DM or to keep the actual portfolio in DM, $u_t = \text{DM}$, then the return is always zero. If the decision has been made to change the portfolio into US-\$ or to keep the actual portfolio in US-\$, $u_t = \text{US-\$}$, then the return is equal to the relative change of the exchange rate weighted with $c_t$. That return is reduced by the transaction costs $\xi$, if the investor has to change into US-\$.

Table 1: The immediate return function.

| $r_t(x_t, c_t, x_{t+1}, u_t)$ | $u_t = \text{DM}$ | $u_t = \text{US-\$}$ |
|---|---|---|
| $c_t \in \text{DM}$ | 0 | $r_t = (x_{t+1}/x_t)(c_t - \xi) - c_t$ |
| $c_t \in \text{US-\$}$ | 0 | $r_t = (x_{t+1}/x_t - 1)c_t$ |

The success of the strategies was tested on a realization (2000 data points) of the exchange rate. The initial investment is 1 DM, at each time step the algorithm has to decide to either change the currency or remain in the present currency.

As a reinforcement learning method, QL has to interact with the environment to learn optimal behavior. Thus, a second set of 2000 data was used to learn the Q-values. The training phase is divided into epochs. Each epoch consists of as many trials as data exist in the training set. At every trial the algorithm looks at $x_t$, chooses randomly a portfolio value $c_t$ and selects a decision. Then the immediate return and the new state is evaluated to apply eq. 3. The Q-values were initialized with zero, the learning rate $\eta$ was 0.1. Convergence was achieved after 4 epochs.

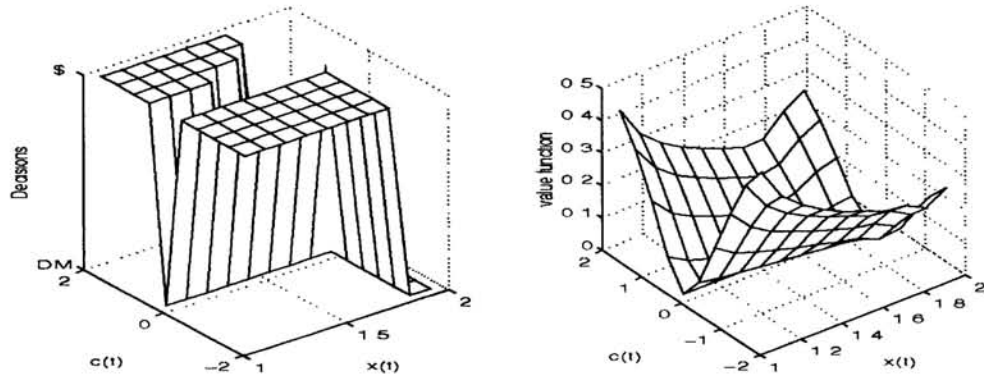

Figure 1: The optimal decisions (left) and value function (right).

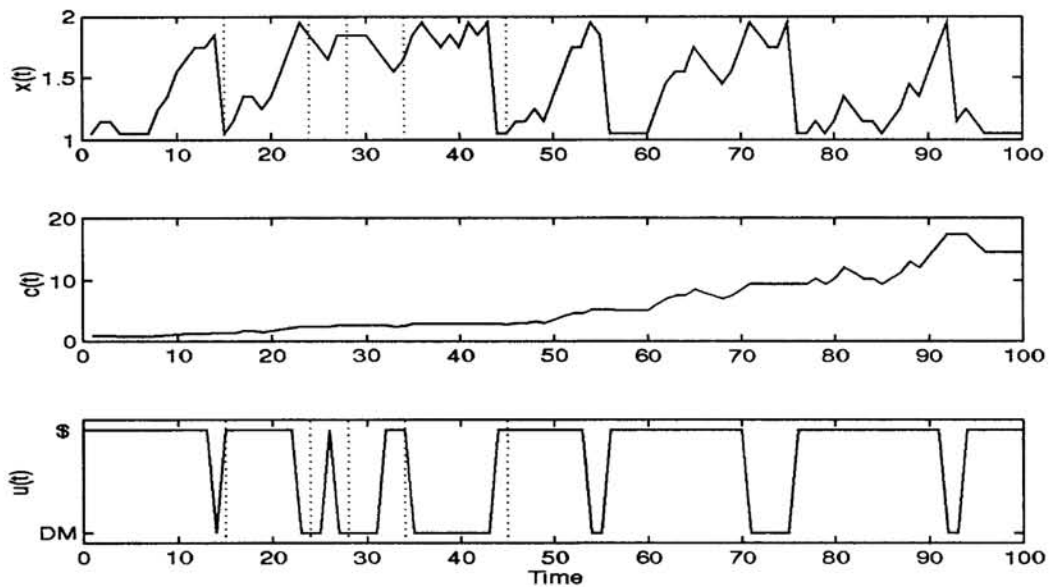

Figure 2: The exchange rate (top), the capital and the decisions (bottom).

To evaluate the solution QL has found, the DP-algorithm from eq. 2 was implemented using the given transition probabilities. The convergence of DP was very fast. Only 5 iterations were needed until the average difference between successive value functions was lower than 0.01. That means 500 updates in comparison to 8000 updates with QL.

The solutions were identical with respect to the resulting policy which is plotted in fig. 1, left. It can clearly be seen, that there is a difference between the policy of a DM-based and a US-\$-based portfolio. If one has already changed the capital to US-\$, then it is advisable to keep the portfolio in US-\$ until the risk gets too high, i. e. $x_t \in \{1.8, 1.9\}$. On the other hand, if $c_t$ is still in DM, the risk barrier moves to lower values depending on the volume of the portfolio. The reason is that the potential gain by an increasing exchange rate has to cover the fixed and variable transaction costs. For very low values of $c_t$, it is forbidden to change even at low $x_t$ because the fixed transaction costs will be higher than any gain. Figure 2 plots the

exchange rate $x_t$, the accumulated capital $c_t$ for 100 days, and the decisions $u_t$.

Let us look at a few interesting decisions. At the beginning, $t = 0$, the portfolio was changed immediately to US-$ and kept there for 13 steps until a drop to low rates $x_t$ became very probable. During the time steps 35-45, the exchange rate oscillated at higher exchange rates. The policy insisted on the DM portfolio, because the risk was too high. In contrary, looking at the time steps 24 to 28, the policy first switched back to DM, then there was a small decrease of $x_t$ which was sufficient to let the investor change again. The following increase justified that decision. The success of the resulting strategy can be easily recognized by the continuous increase of the portfolio. Note, that the ups and downs of the portfolio curve get higher in magnitude at the end because the investor has no risk aversion and always the whole capital is traded.

## 5   Experiment II: German Stock Index DAX

In this section the approach is tested on a real world task: assume that an investor wishes to invest her/his capital into a block of stocks which behaves like the German stock index DAX. We based the benchmark strategy (short: MLP) on a NN model which was build to predict the daily changes of the DAX (for details, see Dichtl, 1995). If the prediction of the next day DAX difference is positive then the capital is invested into DAX otherwise in DM. The input vector of the NN model was carefully optimized for optimal prediction. We used these inputs (the DAX itself and 11 other influencing market variables) as the market description part of the state vector for QL. In order to store the value functions two NNs, one for each action, with 8 nonlinear hidden neurons and one linear output are used.

The data is split into a training (from 2. Jan. 1986 to 31. Dec. 1992) and a test set (from 2. Jan. 1993 to 18. Oct. 1995). The return function is defined in the same way as in section 4 using 0.4% as proportional costs and 0.001 units as fixed costs, which are realistic for financial institutions. The training proceeds as outlined in the previous section with $\eta = 0.001$ for 1000 epochs.

In fig. 3 the development of a reinvested capital is plotted for the optimized (upper line) and the MLP strategy (middle line). The DAX itself is also plotted but with a scaling factor to fit it into the figure (lower line). The resulting policy by QL clearly beats the benchmark strategy because the extra return amounts to 80% at the end of the training period and to 25% at the end of the test phase. A closer look at some statistics can explain the success. The QL policy proposes almost as often as the MLP policy to invest in DAX, but the number of changes from DM to DAX and v. v. is much lower (see table 2). Furthermore, it seems that the QL strategy keeps the capital out of the market if there is no significant trend to follow and the market shows too much volatility (see fig. 3 with straight horizontal lines of the capital development curve indicating no investments). An extensive analysis of the resulting strategy will be the topic of future research.

In a further experiment the NNs which store the Q-values are initialized to imitate the MLP strategy. In some runs the number of necessary epochs were reduced by a factor of 10. But often the QL algorithm took longer to converge because the initialization ignores the input elements which describe the investor's capital and therefore led to a bad starting point in the weight space.

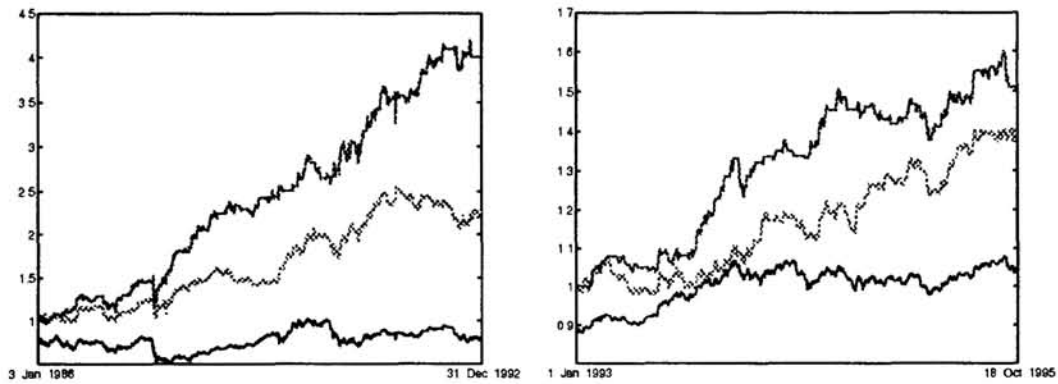

Figure 3: The development of a reinvested capital on the training (left) and test set (right). The lines from top to bottom: QL-strategy, MLP-strategy, scaled DAX.

Table 2: Some statistics of the policies.

|  |  | DAX investments | | position changes | |
|---|---|---|---|---|---|
|  | Data | MLP Policy | QL-Policy | MLP Policy | QL-Policy |
| Training set | 1825 | 1020 | 1005 | 904 | 284 |
| Test set | 729 | 434 | 395 | 344 | 115 |

## 6  Conclusions and Future Work

In this paper, the task of asset allocation/portfolio management was approached by reinforcement learning algorithms. QL was successfully utilized in combination with NNs as value function approximators in a high dimensional state space.

Future work has to address the possibility of several alternative investment opportunities and to clarify the connection to the classical mean-variance approach of professional brokers. The benchmark strategy in the real world experiment is in fact a neuro-fuzzy model which allows the extraction of useful rules after learning. It will be interesting to use that network architecture to approximate the value function in order to achieve a deeper insight in the resulting optimized strategy.

## Footnotes

* Ralph.Neuneier@zfe.siemens.de, http://www.siemens.de/zfe_nn/homepage.html

[1] In the MDP-literature, the return often depends only on the current state $i$, but the theory extends to the case of $r = r(i, j, u(i))$ (see Singh, 1994).

[2]To be more precise, the problem only becomes a multi-stage decision problem if the transaction costs are included in the problem.

### References

Barto A. G., Sutton R. S. and Watkins C. J. C. H. (1989), Learning and Sequential Decision Making, COINS TR 89-95.
Bertsekas D. P. (1987), Dynamic Programming, NY: Wiley.
Singh, P. S. (1993), Learning to Solve Markovian Decision Processes, CMPSCI TR 93-77.
Neuneier R. (1995), Optimal Strategies with Density-Estimating Neural Networks, ICANN 95, Paris.
Brealy, R. A., Myers, S. C. (1991), Principles of Corporate Finance, McGraw-Hill.
Watkins C. J., Dayan, P. (1992), Technical Note: Q-Learning, Machine Learning 8, 3/4.
Elton, E. J. , Gruber, M. J. (1971), Dynamic Programming Applications in Finance, The Journal of Finance, 26/2.
Dichtl, H. (1995), Die Prognose des DAX mit Neuro-Fuzzy, masterthesis, engl. abstract in preparation.